# Bayesian Source Localization with the Multivariate Laplace Prior

**Marcel van Gerven**[1,2]     **Botond Cseke**[1]     **Robert Oostenveld**[2]     **Tom Heskes**[1,2]
[1]Institute for Computing and Information Sciences
[2]Donders Institute for Brain, Cognition and Behaviour
Radboud University Nijmegen
Nijmegen, The Netherlands

## Abstract

We introduce a novel multivariate Laplace (MVL) distribution as a sparsity promoting prior for Bayesian source localization that allows the specification of constraints between and within sources. We represent the MVL distribution as a scale mixture that induces a coupling between source variances instead of their means. Approximation of the posterior marginals using expectation propagation is shown to be very efficient due to properties of the scale mixture representation. The computational bottleneck amounts to computing the diagonal elements of a sparse matrix inverse. Our approach is illustrated using a mismatch negativity paradigm for which MEG data and a structural MRI have been acquired. We show that spatial coupling leads to sources which are active over larger cortical areas as compared with an uncoupled prior.

## 1   Introduction

Electroencephalography (EEG) and magnetoencephalography (MEG) provide an instantaneous and non-invasive measure of brain activity. Let $q$, $p$, and $t$ denote the number of sensors, sources and time points, respectively. Sensor readings $\mathbf{Y} \in \mathbb{R}^{q \times t}$ and source currents $\mathbf{S} \in \mathbb{R}^{p \times t}$ are related by

$$\mathbf{Y} = \mathbf{XS} + \mathbf{E} \tag{1}$$

where $\mathbf{X} \in \mathbb{R}^{q \times p}$ is a lead field matrix that represents how sources project onto the sensors and $\mathbf{E} \in \mathbb{R}^{q \times t}$ represents sensor noise.

Unfortunately, localizing distributed sources is an ill-posed inverse problem that only admits a unique solution when additional constraints are defined. In a Bayesian setting, these constraints take the form of a prior on the sources [3, 19]. Popular choices of prior source amplitude distributions are Gaussian or Laplace priors, whose MAP estimates correspond to minimum norm and minimum current estimates, respectively [18]. Minimum norm estimates produce spatially smooth solutions but are known to suffer from depth bias and smearing of nearby sources. In contrast, minimum current estimates lead to focal source estimates that may be scattered too much throughout the brain volume [9].

In this paper, we take the Laplace prior as our point of departure for Bayesian source localization (instead of using just the MAP estimate). The obvious approach is to assume univariate Laplace priors on individual sources. Here, in contrast, we assume a multivariate Laplace distribution over all sources, which allows sources to be coupled. We show that such a distribution can be represented as a scale mixture [2] that differs substantially from the one presented in [5].

Our representation allows the specification of both spatio-temporal as well as sparsity constraints. Since the posterior cannot be computed exactly, we formulate an efficient expectation propagation

algorithm [12] which allows us to approximate the posterior of interest for very large models. Efficiency arises from the block diagonal form of the approximate posterior covariance matrix due to properties of the scale mixture representation. The computational bottleneck then reduces to computation of the diagonal elements of a sparse matrix inverse, which can be solved through Cholesky decomposition of a sparse matrix and application of the Takahashi equation [17]. Furthermore, moment matching is achieved by one-dimensional numerical integrations. Our approach is evaluated on MEG data that was recorded during an oddball task.

## 2 Bayesian source localization

In a Bayesian setting, the goal of source localization is to estimate the posterior

$$p(\mathbf{S} \mid \mathbf{Y}, \mathbf{X}, \boldsymbol{\Sigma}, \boldsymbol{\Theta}) \propto p(\mathbf{Y} \mid \mathbf{S}, \mathbf{X}, \boldsymbol{\Sigma}) p(\mathbf{S} \mid \boldsymbol{\Theta}) \tag{2}$$

where the likelihood term $p(\mathbf{Y} \mid \mathbf{S}) = \prod_t \mathcal{N}(\mathbf{y}_t \mid \mathbf{X}\mathbf{s}_t, \boldsymbol{\Sigma})$ factorizes over time and $\boldsymbol{\Sigma}$ represents sensor noise. The prior $p(\mathbf{S} \mid \boldsymbol{\Theta})$, with $\boldsymbol{\Theta}$ acting as a proxy for the hyper-parameters, can be used to incorporate (neuroscientific) constraints. For simplicity, we assume independent Gaussian noise with a fixed variance $\sigma^2$, i.e., $\boldsymbol{\Sigma} = \sigma^2 \mathbf{I}$. Without loss of generality, we will focus on one time-point $(\mathbf{y}_t, \mathbf{s}_t)$ only and drop the subscript when clear from context.[1]

The source localization problem can be formulated as a (Bayesian) linear regression problem where the source currents $\mathbf{s}$ play the role of the regression coefficients and rows of the lead field matrix $\mathbf{X}$ can be interpreted as covariates. In the following, we define a multivariate Laplace distribution, represented in terms of a scale mixture, as a convenient prior that incorporates both spatio-temporal and sparsity constraints.

The univariate Laplace distribution

$$\mathcal{L}(s \mid \lambda) \equiv \frac{\lambda}{2} \exp(-\lambda|s|) \tag{3}$$

can be represented as a scale mixture of Gaussians [2], the scaling function being an exponential distribution with parameter $\lambda^2/2$. The scale parameter $\lambda$ controls the width of the distribution and thus the regularizing behavior towards zero. Since the univariate exponential distribution is a $\chi_2^2$ distribution, one can alternatively write

$$\mathcal{L}(s \mid \lambda) = \int du\,dv\, \mathcal{N}\left(s \mid 0, u^2 + v^2\right) \mathcal{N}\left(u \mid 0, 1/\lambda^2\right) \mathcal{N}\left(v \mid 0, 1/\lambda^2\right). \tag{4}$$

Eltoft et al [5] defined the multivariate Laplace distribution as a scale mixture of a multivariate Gaussian given by $\sqrt{z}\boldsymbol{\Sigma}^{1/2}\mathbf{s}$ where $\mathbf{s}$ is a standard normal multivariate Gaussian, $\boldsymbol{\Sigma}$ is a positive definite matrix, and $z$ is drawn from a univariate exponential distribution. The work presented in [11] is based on similar ideas but replaces the distribution on $z$ with a multivariate log-normal distribution.

In contrast, we use an alternative formulation of the multivariate Laplace distribution that couples the variances of the sources rather than the source currents themselves. This is achieved by generalizing the representation in Eq. (4) to the multivariate case. For an uncoupled multivariate Laplace distribution, this generalization reads

$$\mathcal{L}(\mathbf{s} \mid \lambda) = \int d\mathbf{u}\,d\mathbf{v} \prod_i \mathcal{N}\left(s_i \mid 0, u_i^2 + v_i^2\right) \mathcal{N}\left(v_i \mid 0, 1/\lambda^2\right) \mathcal{N}\left(u_i \mid 0, 1/\lambda^2\right) \tag{5}$$

such that each source current $s_i$ gets assigned scale variables $u_i$ and $v_i$. We can interpret the scale variables corresponding to source $i$ as indicators of its relevance: the larger (the posterior estimate of) $u_i^2 + v_i^2$, the more relevant the corresponding source. In order to introduce correlations between sources, we define our multivariate Laplace (MVL) distribution as the following scale mixture:

$$\mathcal{L}(\mathbf{s} \mid \lambda, \mathbf{J}) \equiv \int d\mathbf{u}\,d\mathbf{v} \left(\prod_i \mathcal{N}\left(s_i \mid 0, u_i^2 + v_i^2\right)\right) \mathcal{N}\left(\mathbf{v} \mid \mathbf{0}, \mathbf{J}^{-1}/\lambda^2\right) \mathcal{N}\left(\mathbf{u} \mid \mathbf{0}, \mathbf{J}^{-1}/\lambda^2\right), \tag{6}$$

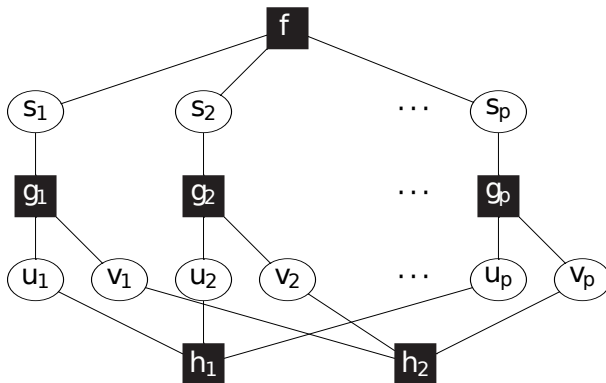

Figure 1: Factor graph representation of Bayesian source localization with a multivariate Laplace prior. The factor $f$ represents the likelihood term $\mathcal{N}\left(\mathbf{y} \mid \mathbf{X}\mathbf{s}, \sigma^2\mathbf{I}\right)$. Factors $g_i$ correspond to the coupling between sources and scales. Factors $h_1$ and $h_2$ represent the (identical) multivariate Gaussians on $\mathbf{u}$ and $\mathbf{v}$ with prior precision matrix $\mathbf{J}$. The $g_i$ are the only non-Gaussian terms and need to be approximated.

where $\mathbf{J}^{-1}$ is a normalized covariance matrix. This definition yields a coupling in the magnitudes of the source currents through their variances. The normalized covariance matrix $\mathbf{J}^{-1}$ specifies the correlation strengths, while $\lambda$ acts as a regularization parameter. Note that this approach is defining the multivariate Laplace with the help of a multivariate exponential distribution [10]. As will be shown in the next section, apart from having a semantics that differs from [5], our scale mixture representation has some desirable characteristics that allow for efficient approximate inference. Based on the above formulation, we define the sparse linear model as

$$p(\mathbf{y}, \mathbf{s} \mid \mathbf{X}, \sigma^2, \lambda, \mathbf{J}) = \mathcal{N}\left(\mathbf{y} \mid \mathbf{X}\mathbf{s}, \sigma^2\mathbf{I}\right) \mathcal{L}\left(\mathbf{s} \mid \lambda, \mathbf{J}\right). \tag{7}$$

The factor graph in Fig. 1 depicts the interactions between the variables in our model.

## 3   Approximate inference

Our goal is to compute posterior marginals for sources $s_i$ as well as scale variables $u_i$ and $v_i$ in order to determine source relevance. These marginals are intractable and we need to resort to approximate inference methods. In this paper we use a deterministic approximate inference method called expectation propagation (EP) [12]. For a detailed analysis of the use of EP in case of the decoupled prior, which is a special case of our MVL prior, we refer to [16]. EP works by iterative minimizations of the Kullback–Leibler (KL) divergence between appropriately chosen distributions in the following way.

We introduce the vector of all latent variables $\mathbf{z} = (\mathbf{s}^T, \mathbf{u}^T, \mathbf{v}^T)^T$. The posterior distribution on $\mathbf{z}$ given the data $\mathbf{y}$ (which is considered fixed and given and therefore omitted in our notation) can be written in the factorized form

$$p(\mathbf{z}) \propto t_0(\mathbf{z}) \prod_i t_i(\mathbf{z}), \tag{8}$$

where $t_0(\mathbf{z}) \propto \mathcal{N}\left(\mathbf{y} \mid \mathbf{X}\mathbf{s}, \sigma^2\mathbf{I}\right) \mathcal{N}\left(\mathbf{v} \mid \mathbf{0}, \mathbf{J}^{-1}/\lambda^2\right) \mathcal{N}\left(\mathbf{u} \mid \mathbf{0}, \mathbf{J}^{-1}/\lambda^2\right)$ and $t_i(\mathbf{z}) = t_i(s_i, u_i, v_i) = \mathcal{N}\left(s_i \mid 0, u_i^2 + v_i^2\right)$. The term $t_0(\mathbf{z})$ is a Gaussian function, i.e., it can be written in the form $\exp(\mathbf{z}^T\mathbf{h}_0 - \mathbf{z}^T\mathbf{K}_0\mathbf{z}/2)$. It factorizes into Gaussian functions of $\mathbf{s}$, $\mathbf{u}$, and $\mathbf{v}$ such that $\mathbf{K}_0$ has a block-diagonal structure. Using EP, we will approximate $p(\mathbf{z})$ with $q(\mathbf{z}) \propto t_0(\mathbf{z}) \prod_i \bar{t}_i(\mathbf{z})$, where the $\bar{t}_i(\mathbf{z})$ are Gaussian functions as well.

Our definition of the MVL distribution leads to several computational benefits. Equation (6) introduces $2p$ auxiliary Gaussian variables $(\mathbf{u}, \mathbf{v})$ that are coupled to the $s_i$'s by $p$ non-Gaussian factors, thus, we have to approximate $p$ terms. The multivariate Laplace distribution defined in [5] introduces one auxiliary variable and couples all the $s_i s_j$ terms to it, therefore, it would lead to $p^2$ non-Gaussian terms to be approximated. Moreover, as we will see below, the a priori independence of $\mathbf{u}$ and $\mathbf{v}$ and

the form of the terms $t_i(\mathbf{z})$ results in an approximation of the posterior with the same block-diagonal structure as that of $t_0(\mathbf{z})$.

In each step, EP updates $\bar{t}_i$ with $\bar{t}_i^*$ by defining $q^{\backslash i} \propto t_0(\mathbf{z}) \prod_{\backslash i} \bar{t}_j$, minimizing KL $\left(t_i q^{\backslash i} \parallel q^*\right)$ with respect to $q^*$ and setting $\bar{t}_i^* \propto q^*/q^{\backslash i}$. It can be shown that when $t_i$ depends only on a subset of variables $\mathbf{z}_i$ (in our case on $\mathbf{z}_i = (s_i, u_i, v_i)$) then so does $\bar{t}_i$. The minimization of the KL divergence then boils down to the minimization of KL $\left(t_i(\mathbf{z}_i)q^{\backslash i}(\mathbf{z}_i) \parallel q^*(\mathbf{z}_i)\right)$ with respect to $q^*(\mathbf{z}_i)$ and $\bar{t}_i$ is updated to $\bar{t}_i^*(\mathbf{z}_i) \propto q^*(\mathbf{z}_i)/q^{\backslash i}(\mathbf{z}_i)$. Minimization of the KL divergence corresponds to moment matching, i.e., $q^*(s_i, u_i, v_i)$ is a Gaussian with the same mean and covariance matrix as $q^i(\mathbf{z}_i) \propto t_i(\mathbf{z}_i)q^{\backslash i}(\mathbf{z}_i)$. So, to update the $i$-th term in a standard application of EP, we would have to compute $q^{\backslash i}(\mathbf{z}_i)$ and could then use a three-dimensional (numerical) integration to compute all first and second moments of $q^i(\mathbf{z}_i)$. Below we will explain how we can exploit the specific characteristics of the MVL to do this more efficiently. For stability, we use a variant of EP, called power EP [13], where $q^{\backslash i} \propto \bar{t}_i^{(1-\alpha)} \prod_{\backslash i} \bar{t}_j$ and KL $\left(t_i^\alpha q^{\backslash i} \parallel q^*\right)$ with $\alpha \in (0, 1]$ is minimized. The above explanation of standard EP corresponds to $\alpha = 1$. In the following we will give the formulas for general $\alpha$.

We will now work out the EP update for the $i$-th term approximation in more detail to show by induction that $\bar{t}_i(s_i, u_i, v_i)$ factorizes into independent terms for $s_i$, $u_i$, and $v_i$. Since $u_i$ and $v_i$ play exactly the same role, it is also easy to see that the term approximation is always symmetric in $u_i$ and $v_i$. Let us suppose that $q(s_i, u_i, v_i)$ and consequently $q^{\backslash i}(s_i, u_i, v_i)$ factorizes into independent terms for $s_i$, $u_i$, and $v_i$, e.g., we can write

$$q^{\backslash i}(s_i, u_i, v_i) = \mathcal{N}(s_i \mid m_i, \sigma_i^2)\mathcal{N}(u_i \mid 0, \nu_i^2)\mathcal{N}(v_i \mid 0, \nu_i^2). \tag{9}$$

By initializing $\bar{t}_i(s_i, u_i, v_i) = 1$, we have $q(\mathbf{z}) \propto t_0(\mathbf{z})$ and the factorization of $q^{\backslash i}(s_i, u_i, v_i)$ follows directly from the factorization of $t_0(\mathbf{z})$ into independent terms for $\mathbf{s}$, $\mathbf{u}$, and $\mathbf{v}$. That is, for the first EP step, the factorization can be guaranteed. To obtain the new term approximation, we have to compute the moments of the distribution $q^i(s_i, u_i, v_i) \propto \mathcal{N}(s_i \mid 0, u_i^2 + v_i^2)^\alpha q^{\backslash i}(s_i, u_i, v_i)$, which, by regrouping terms, can be written in the form $q^i(s_i, u_i, v_i) = q^i(s_i \mid u_i, v_i)q^i(u_i, v_i)$ with

$$q^i(s_i \mid u_i, v_i) \quad \propto \quad \mathcal{N}\left(s_i \mid \frac{m_i(u_i^2 + v_i^2)}{\alpha\sigma_i^2 + u_i^2 + v_i^2}, \frac{\sigma_i^2(u_i^2 + v_i^2)}{\alpha\sigma_i^2 + u_i^2 + v_i^2}\right) \tag{10}$$

$$q^i(u_i, v_i) \quad \propto \quad \left(u_i^2 + v_i^2\right)^{(1-\alpha)/2} \mathcal{N}\left(\sqrt{\alpha}m_i \mid 0, \alpha\sigma_i^2 + u_i^2 + v_i^2\right)$$
$$\times \mathcal{N}(u_i \mid 0, \nu_i^2)\mathcal{N}(v_i \mid 0, \nu_i^2). \tag{11}$$

Since $q^i(u_i, v_i)$ only depends on $u_i^2$ and $v_i^2$ and is thus invariant under sign changes of $u_i$ and $v_i$, we must have E $[u_i]$ = E $[v_i]$ = 0, as well as E $[u_iv_i]$ = 0. Because of symmetry, we further have E $\left[u_i^2\right]$ = E $\left[v_i^2\right]$ = (E $\left[u_i^2\right]$ + E $\left[v_i^2\right]$)/2. Since $q^i(u_i, v_i)$ can be expressed as a function of $u_i^2 + v_i^2$ only, this variance can be computed from (11) using one-dimensional Gauss-Laguerre numerical quadrature [15]. The first and second moments of $s_i$ conditioned upon $u_i$ and $v_i$ follow directly from (10). Because both (10) and (11) are invariant under sign changes of $u_i$ and $v_i$, we must have E $[s_iu_i]$ = E $[s_iv_i]$ = 0. Furthermore, since the conditional moments again depend only on $u_i^2 + v_i^2$, also E $[s_i]$ and E $\left[s_i^2\right]$ can be computed with one-dimensional Gauss-Laguerre integration. Summarizing, we have shown that if the old term approximations factorize into independent terms for $s_i$, $u_i$, and $v_i$, the new term approximation after an EP update, $\bar{t}_i^*(s_i, u_i, v_i) \propto q^*(s_i, u_i, v_i)/q^{\backslash i}(s_i, u_i, v_i)$, must do the same. Furthermore, given the cavity distribution $q^{\backslash i}(s_i, u_i, v_i)$, all required moments can be computed using one-dimensional numerical integration.

The crucial observation here is that the terms $t_i(s_i, u_i, v_i)$ introduce dependencies between $s_i$ and $(u_i, v_i)$, as expressed in Eqs. (10) and (11), but do not lead to correlations that we have to keep track of in a Gaussian approximation. This is not specific to EP, but a consequence of the symmetries and invariances of the exact distribution $p(\mathbf{s}, \mathbf{u}, \mathbf{v})$. That is, also when the expectations are taken with respect to the exact $p(\mathbf{s}, \mathbf{u}, \mathbf{v})$ we have E $[u_i]$ = E $[v_i]$ = E $[u_iv_i]$ = E $[s_iu_i]$ = E $[s_iv_i]$ = 0 and E $\left[u_i^2\right]$ = E $\left[v_i^2\right]$. The variance of the scales E $\left[u_i^2 + v_i^2\right]$ determines the amount of regularization on the source parameter $s_i$ such that large variance implies little regularization.

Last but not least, contrary to conventional sequential updating, we choose to update the terms $\bar{t}_i$ in parallel. That is, we compute all $q^{\backslash i}$s and update all terms simultaneously. Calculating $q^{\backslash i}(s_i, u_i, v_i)$

requires the computation of the marginal moments $q(s_i)$, $q(u_i)$ and $q(v_i)$. For this, we need the diagonal elements of the inverse of the precision matrix $\mathbf{K}$ of $q(\mathbf{z})$. This precision matrix has the block-diagonal form

$$\mathbf{K} = \begin{bmatrix} \mathbf{X}^T\mathbf{X}/\sigma^2 + \mathbf{K}_s & \mathbf{0} & \mathbf{0} \\ \mathbf{0} & \lambda^2\mathbf{J} + \mathbf{K}_u & \mathbf{0} \\ \mathbf{0} & \mathbf{0} & \lambda^2\mathbf{J} + \mathbf{K}_v \end{bmatrix} \tag{12}$$

where $\mathbf{J}$ is a sparse precision matrix which determines the coupling, and $\mathbf{K}_s$, $\mathbf{K}_u$, and $\mathbf{K}_v = \mathbf{K}_u$ are diagonal matrices that contain the contributions of the term approximations. We can exploit the low-rank representation of $\mathbf{X}^T\mathbf{X}/\sigma^2 + \mathbf{K}_s$ to compute its inverse using the Woodbury formula [7]. The diagonal elements of the inverse of $\lambda^2\mathbf{J} + \mathbf{K}_u$ can be computed efficiently via sparse Cholesky decomposition and the Takahashi equation [17]. By updating the term approximations in parallel, we only need to perform these operations once per parallel update.

## 4  Experiments

Returning to the source localization problem, we will show that the MVL prior can be used to induce constraints on the source estimates. To this end, we use a dataset obtained for a mismatch negativity experiment (MMN) [6]. The MMN is the negative component of the difference between responses to normal and deviant stimuli within an oddball paradigm that peaks around 150 ms after stimulus onset. In our experiment, the subject had to listen to normal (500 Hz) and deviant (550 Hz) tones, presented for 70 ms. Normal tones occurred 80% of the time, whereas deviants occurred 20% of the time. A total of 600 trials was acquired.

Data was acquired with a CTF MEG System (VSM MedTech Ltd., Coquitlam, British Columbia, Canada), which provides whole-head coverage using 275 DC SQUID axial gradiometers. A realistically shaped volume conduction model was constructed based on the individual's structural MRI [14]. The brain volume was discretized to a grid with a 0.75 cm resolution and the lead field matrix was calculated for each of the 3863 grid points according to the head position in the system and the forward model. The lead field matrix is defined for the three $x$, $y$, and $z$ orientations in each of the source locations and was normalized to correct for depth bias. Consequently, the lead field matrix $\mathbf{X}$ is of size $275 \times 11589$. The $275 \times 1$ observation vector $\mathbf{y}$ was rescaled to prevent issues with numerical precision.

In the next section, we compare source estimates for the MMN difference wave that have been obtained when using either a decoupled or a coupled MVL prior. For ease of exposition, we focus on a spatial prior induced by the coupling of neighboring sources. In order to demonstrate the effect of the spatial prior, we assume a fixed regularization parameter $\lambda$ and fixed noise variance $\sigma^2$, as estimated by means of the L curve criterion [8]. Differences in the source estimates will therefore arise only from the form of the $11589 \times 11589$ sparse precision matrix $\mathbf{J}$. The first estimate is obtained by assuming that there is no coupling between elements of the lead field matrix, such that $\mathbf{J} = \mathbf{I}$. This gives a Bayesian formulation of the minimum current estimate [18]. The second estimate is obtained by assuming a coupling between neighboring sources $i$ and $j$ within the brain volume with fixed strength $c$. This coupling is specified through the *unnormalized* precision matrix $\hat{\mathbf{J}}$ by assuming $\hat{J}_{i_x,j_x} = \hat{J}_{i_y,j_y} = \hat{J}_{i_z,j_z} = -c$ while diagonal elements $\hat{J}_{ii}$ are set to $1 - \sum_{j \neq i} \hat{J}_{ij}$.[2] This prior dictates that the magnitude of the variances of the source currents are coupled between sources.

For the coupling strength $c$, we use correlation as a guiding principle. Recall that the unnormalized precision matrix $\hat{\mathbf{J}}$ in the end determines the correlations (of the variances) between sources. Specifically, correlation between sources $s_i$ and $s_j$ is given by

$$r_{ij} = \left(\hat{\mathbf{J}}^{-1}\right)_{ij} / \left(\hat{\mathbf{J}}^{-1}\right)_{ii}^{\frac{1}{2}} \left(\hat{\mathbf{J}}^{-1}\right)_{jj}^{\frac{1}{2}}. \tag{13}$$

For example, using $c = 10$, we would obtain a correlation coefficient of $r_{i,i+1} = 0.78$. Note that this also leads to more distant sources having non-zero correlations. The positive correlation between

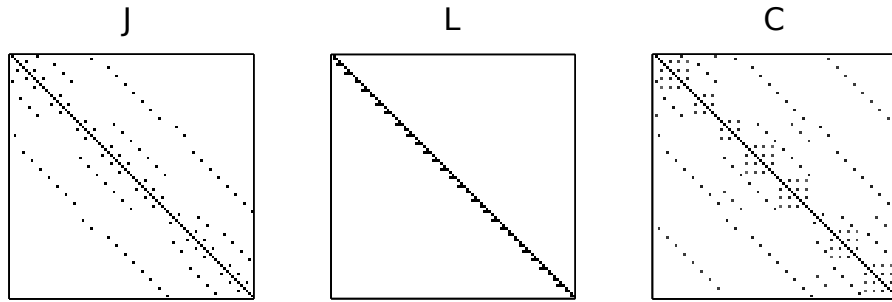

Figure 2: Spatial coupling leads to the normalized precision matrix $\mathbf{J}$ with coupling of neighboring source orientations in the $x$, $y$, and $z$ directions. The (reordered) matrix $\mathbf{L}$ is obtained from the Cholesky decomposition of $\mathbf{J}$. The correlation matrix $\mathbf{C}$ shows the correlations between the source orientations. For the purpose of demonstration, we show matrices using a very coarse discretization of the brain volume.

neighboring sources is motivated by the notion that we expect neighboring sources to be similarly though not equivalently involved for a given task. Evidently, the desired correlation coefficient also depends on the resolution of the discretized brain volume.

Figure 2 demonstrates how a chosen coupling leads to a particular structure of $\mathbf{J}$, where irregularities in $\mathbf{J}$ are caused by the structure of the imaged brain volume. The figure also shows the computational bottleneck of our algorithm, which is to compute diagonal elements of $\mathbf{J}^{-1}$. This can be solved by means of the Takahashi equation which operates on the matrix $\mathbf{L}$ that results from a sparse Cholesky decomposition. The block diagonal structure of $\mathbf{L}$ arises from a reordering of rows and columns using, for instance, the *amd* algorithm [1]. The correlation matrix $\mathbf{C}$ shows the correlations between the sources induced by the structure of $\mathbf{J}$. Zeros in the correlation matrix arise from the independence between source orientations $x$, $y$, and $z$.

## 5 Results

Figure 3 depicts the difference wave that was obtained by subtracting the trial average for standard tones from the trial average for deviant tones. A negative deflection after 100 ms is clearly visible. The event-related field indicates patterns of activity at central channels in both hemispheres. These

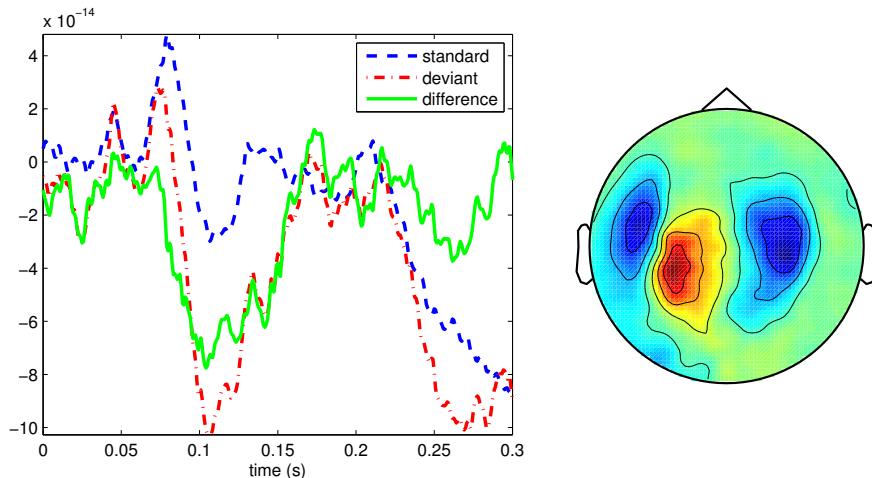

Figure 3: Evolution of the difference wave at right central sensors and event-related field of the difference wave 125 ms after cue onset.

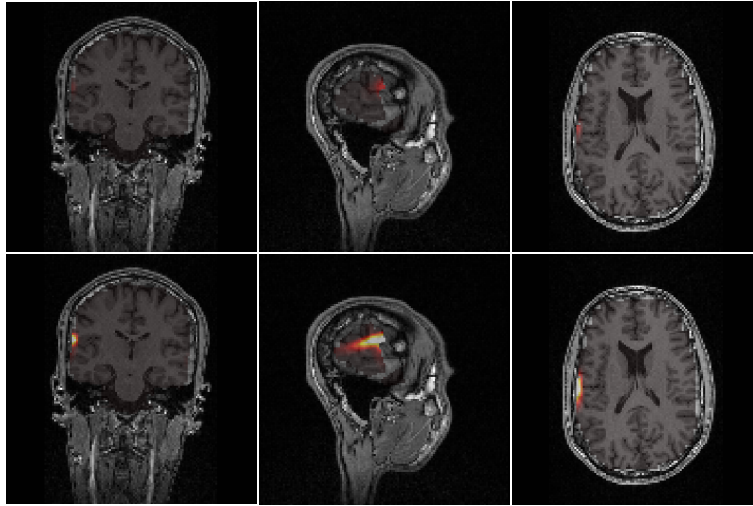

Figure 4: Source estimates using a decoupled prior (top) or a coupled prior (bottom). Plots are centered on the left temporal source.

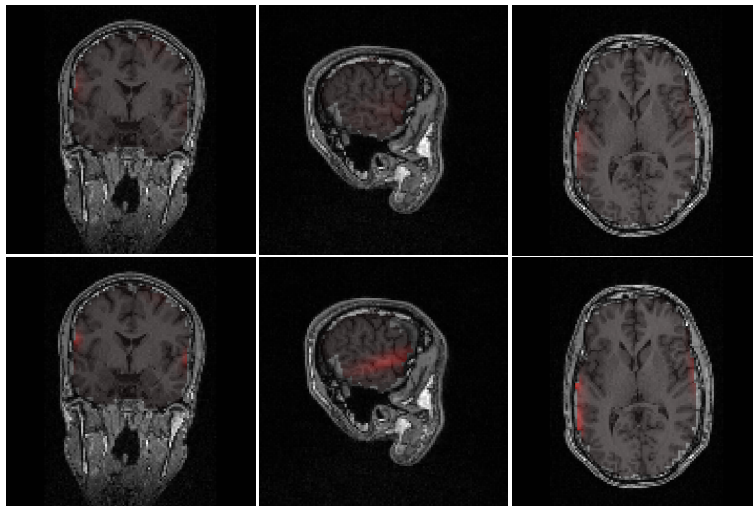

Figure 5: Relative variance using a decoupled prior (top) or a coupled prior (bottom). Plots are centered on the right temporal source.

findings are consistent with the mismatch negativity literature [6]. We now proceed to localizing the sources of the activation induced by mismatch negativity.

Figure 4 depicts the localized sources when using either a decoupled MVL prior or a coupled MVL prior. The coupled spatial prior leads to stronger source currents that are spread over a larger brain volume. MVL source localization has correctly identified the source over left temporal cortex but does not capture the source over right temporal cortex that is also hypothesized to be present (cf. Fig. 3). Note however that the source estimates in Fig. 4 represent estimated mean power and thus do not capture the full posterior over the sources.

Differences between the decoupled and the coupled prior become more salient when we look at the relative variance of the auxiliary variables as shown in Fig. 5. Relative variance is defined here as posterior variance minus prior variance of the auxiliary variables, normalized to be between zero and one. This measure indicates the change in magnitude of the variance of the auxiliary variables, and thus indirectly that of the sources via Eq. (6). Since only sources with non-zero contributions should have high variance, this measure can be used to indicate the relevance of a source. Figure 5

shows that temporal sources in both left and right hemispheres are relevant. The relevance of the temporal source in the right hemisphere becomes more pronounced when using the coupled prior.

## 6 Discussion

In this paper, we introduced a multivariate Laplace prior as the basis for Bayesian source localization. By formulating this prior as a scale mixture we were able to approximate posteriors of interest using expectation propagation in an efficient manner. Computation time is mainly influenced by the sparsity structure of the precision matrix $\mathbf{J}$ which is used to specify interactions between sources by coupling their variances. We have demonstrated the feasibility of our approach using a mismatch negativity dataset. It was shown that coupling of neighboring sources leads to source estimates that are somewhat more spatially smeared as compared with a decoupled prior. Furthermore, visualization of the relative variance of the auxiliary variables gave additional insight into the relevance of sources.

Contrary to the MAP estimate (i.e., the minimum current estimate), our Bayesian estimate does not exactly lead to sparse posteriors given a finite amount of data. However, posterior marginals can still be used to exclude irrelevant sources since these will typically have a mean activation close to zero with small variance. In principle, we could force our posteriors to become more MAP-like by replacing the likelihood term with $\mathcal{N}\left(\mathbf{y} \mid \mathbf{Xs}, \sigma^2\mathbf{I}\right)^{1/T}$ in the limit $T \rightarrow 0$. From the Bayesian point of view, one may argue whether taking this limit is fair. In any case, given the inherent uncertainty in our estimates we favor the representation in terms of (non-sparse) posterior marginals.

Note that it is straightforward to impose other constraints since this only requires the specification of suitable interactions between sources through $\mathbf{J}$. For instance, the spatial prior could be made more realistic by taking anatomical constraints into account or by the inclusion of coupling between sources over time. Other constraints that can be implemented with our approach are the coupling of individual orientations within a source, or even the coupling of source estimates between different subjects. Coupling of source orientations has been realized before in [9] through an $\ell_1/\ell_2$ norm, although not using a fully Bayesian approach. In future work, we aim to examine the effect of the proposed priors and optimize the regularization and coupling parameters via empirical Bayes [4]. Other directions for further research are inclusion of the noise variance in the optimization procedure and dealing with the depth bias that often arises in distributed source models in a more principled way.

In [11], fields of Gaussian scale mixtures were used for modeling the statistics of wavelet coefficients of photographics images. Our approach differs in two important aspects. To obtain a generalization of the univariate Laplace distribution, we used a multivariate exponential distribution of the scales, to be compared with the multivariate log-normal distribution in [11]. The Laplace distribution has the advantage that it is the most sparsifying prior that, in combination with a linear model, still leads to a unimodal posterior [16]. Furthermore, we described an efficient method for approximating marginals of interest whereas in [11] an iterative coordinate-ascent method was used to compute the MAP solution. Since (the efficiency of) our method for approximate inference only depends on the sparsity of the multivariate scale distribution, and not on its precise form, it should be feasible to compute approximate marginals for the model presented in [11] as well.

Concluding, we believe the scale mixture representation of the multivariate Laplace distribution to be a promising approach to Bayesian distributed source localization. It allows a wide range of constraints to be included and, due to the characteristics of the scale mixture, posteriors can be approximated efficiently even for very large models.

**Acknowledgments**

The authors gratefully acknowledge the support of the Dutch technology foundation STW (project number 07050) and the BrainGain Smart Mix Programme of the Netherlands Ministry of Economic Affairs and the Netherlands Ministry of Education, Culture and Science. Tom Heskes is supported by Vici grant 639.023.604.

## Footnotes

[1]Multiple time-points can be incorporated by vectorizing $\mathbf{Y}$ and $\mathbf{S}$, and augmenting $\mathbf{X}$.

[2]The normalized precision matrix is obtained through $\mathbf{J} = \text{diag}(\hat{\mathbf{J}}^{-1})^{\frac{1}{2}} \hat{\mathbf{J}} \text{diag}(\hat{\mathbf{J}}^{-1})^{\frac{1}{2}}$.

# References

[1] P. R. Amestoy, T. A. Davis, and I. S. Duff. Algorithm 837: Amd, an approximate minimum degree ordering algorithm. *ACM Transactions on Mathematical Software*, 30(3):381–388, 2004.

[2] D. F. Andrews and C. L. Mallows. Scale mixtures of normal distributions. *Journal of the Royal Statistical Society, Series B*, 36(1):99–102, 1974.

[3] S. Baillet and L. Garnero. A Bayesian approach to introducing anatomo-functional priors in the EEG/MEG inverse problem. *IEEE Transactions on Biomedical Engineering*, 44(5):374–385, 1997.

[4] J. M. Bernardo and J. F. M. Smith. *Bayesian Theory*. Wiley, 1994.

[5] T. Eltoft, T. Kim, and T. Lee. On the multivariate Laplace distribution. *IEEE Signal Processing Letters*, 13(5):300–303, 2006.

[6] M. I. Garrido, J. M. Kilner, K. E. Stephan, and K. J. Friston. The mismatch negativity: A review of underlying mechanisms. *Clinical Neurophysiology*, 120:453–463, 2009.

[7] G. Golub and C. van Loan. *Matrix Computations*. John Hopkins University Press, Baltimore, MD, 3rd edition, 1996.

[8] P. C. Hansen. *Rank-Deficient and Discrete Ill-Posed Problems: Numerical Aspects of Linear Inversion*. Monographs on Mathematical Modeling and Computation. Society for Industrial Mathematics, 1987.

[9] S. Haufe, V. V. Nikulin, A. Ziehe, K.-R. Müller, and G. Nolte. Combining sparsity and rotational invariance in EEG/MEG source reconstruction. *NeuroImage*, 42(2):726–738, 2008.

[10] N. T. Longford. Classes of multivariate exponential and multivariate geometric distributions derived from Markov processes. In H. W. Block, A. R. Sampson, and T. H. Savits, editors, *Topics in statistical dependence*, volume 16 of *IMS Lecture Notes Monograph Series*, pages 359–369. IMS Business Office, Hayward, CA, 1990.

[11] S. Lyu and E. P. Simoncelli. Statistical modeling of images with fields of Gaussian scale mixtures. In B. Schölkopf, J. Platt, and T. Hoffman, editors, *Advances in Neural Information Processing Systems 19*, pages 945–952. MIT Press, Cambridge, MA, 2007.

[12] T. Minka. Expectation propagation for approximate Bayesian inference. In J. Breese and D. Koller, editors, *Proceedings of the Seventeenth Conference on Uncertainty in Artificial Intelligence*, pages 362–369. Morgan Kaufmann, 2001.

[13] T. Minka. Power EP. Technical report, Microsoft Research, Cambridge, 2004.

[14] G. Nolte. The magnetic lead field theorem in the quasi-static approximation and its use for magnetoencephalography forward calculation in realistic volume conductors. *Physics in Medicine & Biology*, 48(22):3637–3652, 2003.

[15] W. H. Press, S. A. Teukolsky, W. T. Vetterling, and B. P. Flannery. *Numerical Recipes in C*. Cambridge University Press, 3rd edition, 2007.

[16] M. W. Seeger. Bayesian inference and optimal design for the sparse linear model. *Journal of Machine Learning Research*, 9:759–813, 2008.

[17] K. Takahashi, J. Fagan, and M. S. Chen. Formation of a sparse bus-impedance matrix and its application to short circuit study. In *8th IEEE PICA Conference*, pages 63–69, Minneapolis, MN, 1973.

[18] K. Uutela, M. Hämäläinen, and E. Somersalo. Visualization of magnetoencephalographic data using minimum current estimates. *NeuroImage*, 10:173–180, 1999.

[19] D. Wipf and S. Nagarajan. A unified Bayesian framework for MEG/EEG source imaging. *NeuroImage*, 44(3):947–966, 2009.

